# Augmented Rescorla-Wagner and Maximum Likelihood estimation.

**Alan Yuille**
Department of Statistics
University of California at Los Angeles
Los Angeles, CA 90095
yuille@stat.ucla.edu

## Abstract

We show that linear generalizations of Rescorla-Wagner can perform Maximum Likelihood estimation of the parameters of all generative models for causal reasoning. Our approach involves augmenting variables to deal with conjunctions of causes, similar to the agumented model of Rescorla. Our results involve genericity assumptions on the distributions of causes. If these assumptions are violated, for example for the Cheng causal power theory, then we show that a linear Rescorla-Wagner can estimate the parameters of the model up to a nonlinear transformtion. Moreover, a nonlinear Rescorla-Wagner is able to estimate the parameters directly to within arbitrary accuracy. Previous results can be used to determine convergence and to estimate convergence rates.

## 1  Introduction

It is important to understand the relationship between the Rescorla-Wagner (RW) algorithm [1,2] and theories of learning based on maximum likelihood (ML) estimation of the parameters of generative models [3,4,5]. The Rescorla-Wagner algorithm has been shown to account for many experimental findings. But maximum likelihood offers the promise of a sound statistical basis including the ability to learn sophisticated probabilistic models for causal learning [6,7,8].

Previous work, summarized in section (2), showed a direct relationship between the basic Rescorla-Wagner algorithm and maximum likelihood for the $\Delta P$ model of causal learning [4,9]. More recently, a generalization of Rescorla-Wagner was shown to perform maximum likelihood estimation for both the $\Delta P$ and the noisy-or models [10]. *Throughout the paper, we follow the common practice of studying the convergence of the expected value of the weights and ignoring the fluctuations. The size of these fluctuations can be calculated analytically and precise convergence quantified* [10].

In this paper, we greatly extend the connections between Rescorla-Wagner and ML estimation. We show that two classes of generalized Rescorla-Wagner algorithms can perform ML estimation for all generative models provided genericity assumptions on the causes are satisfied. These generalizations include *augmenting the set of variables to represent conjunctive causes* and are related to the augmented Rescorla-Wagner algorithm [2].

We also analyze the case where the genericity assumption breaks down and pay particular attention to Chengs' causal power model [4,5]. We demonstrate that Rescorla-Wagner can perform ML estimation for this model up to a nonlinear transformation of the model parameters (i.e. Rescorla-Wagner does ML but in a different coordinate system). We sketch how a nonlinear Rescorla-Wagner can estimate the parameters directly.

Convergence analysis from previous work [10] can be directly applied to these new Rescorla-Wagner algorithms. This gives convergence conditions and put bounds on the convergence rate. The analysis assumes that the data consists of i.i.d. samples from the (unknown) causal distribution. But the results can also be applied in the piecewise iid case (such as forward and backward blocking [11]).

## 2    Summary of Previous Work

We summarize pervious work relating maximum likelihood estimation of generative models with the Rescorla-Wagner algorithm [4,9,10]. This work assumes that there is a binary-valued event $E$ which can be caused by one or more of two binary-valued causes $C_1, C_2$. The $\Delta P$ and Noisy-or theories use generative models of form:

$$P_{\Delta P}(E = 1|C_1, C_2, \omega_1, \omega_2) = \omega_1 C_1 + \omega_2 C_2 \tag{1}$$

$$P_{Noisy-or}(E = 1|C_1, C_2, \omega_1, \omega_2) = \omega_1 C_1 + \omega_2 C_2 - \omega_1 \omega_2 C_1 C_2, \tag{2}$$

where $\{\omega_1, \omega_2\}$ are the model parameters.

The training data consists of examples $\{E^\mu, C_1^\mu, C_2^\mu\}$. The parameters $\{\omega_1, \omega_2\}$ are estimated by Maximum Likelihood

$$\{\omega_1^*, \omega_2^*\} = \arg \max_{\{\omega_1, \omega_2\}} \prod_\mu P(E^\mu|C_1^\mu, C_2^\mu; \omega_1, \omega_2) P(C_1^\mu, C_2^\mu), \tag{3}$$

where $P(C_1, C_2)$ is the distribution on the causes. It is independent of $\{\omega_1, \omega_2\}$ and does not affect the Maximum Likelihood estimation, except for some non-generic cases to be discussed in section (5).

An alternative approach to learning causal models is the Rescorla-Wagner algorithm which updates weights $V_1, V_2$ as follows:

$$V_1^{t+1} = V_1^t + \Delta V_1^t, \quad V_2^{t+1} = V_2^t + \Delta V_2^t, \tag{4}$$

where the update rule $\Delta V$ can take forms like:

$$\Delta V_1 = \alpha_1 C_1(E - C_1 V_1 - C_2 V_2), \quad \Delta V_2 = \alpha_2 C_2(E - C_1 V_1 - C_2 V_2), \text{ basic rule} \tag{5}$$

$$\Delta V_1 = \alpha_1 C_1(1 - C_2)(E - V_1), \quad \Delta V_2 = \alpha_2 C_2(1 - C_1)(E - V_2), \text{ variant rule.} \tag{6}$$

It is known that if the basic update rule (5) is used then the weights converge to the ML estimates of the parameters $\{\omega_1, \omega_2\}$ provided the data is generated by the $\Delta P$ model (1) [4,9] (but not for the noisy-or model).

If the variant update rule (6) is used, then the weights converge to the parameters $\{\omega_1, \omega_2\}$ of the $\Delta P$ model *or* the noisy-or model (2) depending on which model generates the data [10].

## 3    Basic Ingredients

This section describes three basic ingredients of this work: (i) the generative models, (ii) maximum likelihood, and (iii) the generalized Rescorla-Wagner algorithms.

**Representing the generative models.**

We represent the distribution $P(E|\vec{C};\vec{\alpha})$ by the function:

$$P(E = 1|\vec{C};\vec{\alpha}) = \sum_i \alpha_i h_i(\vec{C}), \tag{7}$$

where the $\{h_i(\vec{C})\}$ are a set of basis functions and the $\{\alpha_i\}$ are parameters. If the dimension of $\vec{C}$ is $n$, then the number of basis functions is $2^n$. All distributions of binary variables can be represented in this form.

For example, if $n = 2$ we can use the basis:

$$h_1(\vec{C}) = 1, h_2(\vec{C}) = C_1, h_3(\vec{C}) = C_2, h_4(\vec{C}) = C_1 C_2, \tag{8}$$

Then the noisy-or model $P(E = 1|C_1, C_2) = \omega_1 C_1 + \omega_2 C_2 - \omega_1 \omega_2 C_1 C_2$ corresponds to setting $\alpha_1 = 0, \alpha_2 = \omega_1, \alpha_3 = \omega_2, \alpha_4 = -\omega_1 \omega_2$.

**Data Generation Assumption and Maximum Likelihood**

We assume that the observed data $\{E^\mu, \vec{C}^\mu : \mu \in \Lambda\}$ are i.i.d. samples from $P(E|\vec{C})P(\vec{C})$. It is possible to adapt our results to cases where the data is piecewise i.i.d., such as blocking experiments, but we have no space to describe this here.

Maximum Likelihood (ML) estimates the $\vec{\alpha}$ by solving:

$$\vec{\alpha}^* = \arg \min_{\vec{\alpha}} - \sum_{\mu \in \Lambda} \log\{P(E^\mu|\vec{C}^\mu;\vec{\alpha})P(\vec{C}^\mu)\} = \arg \min_{\vec{\alpha}} - \sum_{\mu \in \Lambda} \log P(E^\mu|\vec{C}^\mu;\vec{\alpha}). \tag{9}$$

Observe that the estimate of $\vec{\alpha}$ is independent of $P(\vec{C})$ provided the distribution is generic. Important non-generic cases are treated in section (5).

**Generalized Rescorla-Wagner**.

The Rescorla-Wagner (RW) algorithm updates weights $\{V_i : i = 1, ..., n\}$ by a discrete iterative algorithm:

$$V_i^{t+1} = V_i^t + \Delta V_i^t, \;\; i = 1, ..., n. \tag{10}$$

We assume a generalized form:

$$\Delta V_i = \sum_j V_j f_{ij}(\vec{C}) + E g_i(\vec{C}), \;\; i, j = 1, ..., n \tag{11}$$

for functions $\{f_{ij}(\vec{C})\}, \{g_i(\vec{C})\}$. It is easy to see that equations (5,6) are special cases.

## 4 Theoretical Results

We now gives sufficient conditions which ensure that the only fixed points of generalized Rescorla-Wagner correspond to ML estimates of the parameters $\vec{\alpha}$ of generative models $P(E|\vec{C},\vec{\alpha})$. We then obtain two classes of generalized Rescorla-Wagner which satisfy these conditions. For one class, convergence to the fixed points follow directly. For the other class we need to adapt results from [10] to guarantee convergence to the fixed points. Our results assume genericity conditions on the distribution $P(\vec{C})$ of causes. We relax these conditions in section (5).

The number of weights $\{V_i\}$ used by the Rescorla-Wagner algorithm is equal to the number of parameters $\{\alpha_i\}$ that specify the model. But many weights will remain zero unless conjunctions of causes occur, see section (6).

**Theorem 1.** *A sufficient condition for generalized Rescorla-Wagner (11), to have a unique fixed point at the maximum likelihood estimates of the parameters of a generative model $P(E|\vec{C};\vec{\alpha})$ (7), is that $< f_{ij}(\vec{C}) >_{P(\vec{C})} = - < g_i(\vec{C})h_j(\vec{C}) >_{P(\vec{C})} \ \forall \, i,j$ and the matrix $< f_{ij}(\vec{C}) >_{P(\vec{C})}$ is invertible.*

Proof. *We calculate the expectation $< \Delta V_i >_{P(E|\vec{C})P(\vec{C})}$. This is zero if, and only if, $\sum_j V_j < f_{ij}(\vec{C}) >_{P(\vec{C})} + \sum_j \alpha_j < g_i(\vec{C})h_j(\vec{C}) >_{P(\vec{C})} = 0$. The result follows.*

We use notation that $< \, . \, >_{P(\vec{C})}$ is the expectation with respect to the probability distribution $P(\vec{C})$ on the causes. For example, $< f_{ij}(\vec{C}) >_{P(\vec{C})} = \sum_{\vec{C}} P(\vec{C}) f_{ij}(\vec{C})$. Hence the requirement that the matrix $< f_{ij}(\vec{C}) >_{P(\vec{C})}$ is invertible usually requires that $P(\vec{C})$ is generic. See examples in sections (4.1,4.2). Convergence may still occur if the matrix $< f_{ij}(\vec{C}) >_{P(\vec{C})}$ is non-invertible. Linear combinations of the weights will remained fixed (in the directions of the zero eigenvectors of the matrix) and the remaining linear co,mbinations will converge.

Additional conditions to ensure convergence to the fixed point, and to determine the convergence rate, can be found using Theorems 3,4,5 in [10].

## 4.1 Generalized RW class I

We now give prove a corollary of Theorem 1 which will enable us to obtain our first class of generalized RW algorithms.

**Corollary 1**. *A sufficient condition for generalized RW to have fixed points at ML estimates of the model parameters is $f_{ij}(\vec{C}) = -h_i(\vec{C})h_j(\vec{C})$, $g_i(\vec{C}) = h_i(\vec{C}) \ \forall \, i,j$ and the matrix $< h_i(\vec{C})h_j(\vec{C}) >_{P(\vec{C})}$ is invertible. Moreover, convergence to the fixed point is guaranteed.*

Proof. *Direct verification. Convergence to the fixed point follows from the gradient descent nature of the algorithm, see equation (12).*

These conditions define generalized RW class I (GRW-I) which is a natural extension of basic Rescorla-Wagner (5):

$$\Delta V_i = h_i(\vec{C})\{E - \sum_j h_j(\vec{C})V_j\} = -\frac{\partial}{\partial V_i}(E - \sum_j h_j(\vec{C})V_j)^2, \ \ i = 1,...,n \quad (12)$$

This GRW-I algorithm ia guaranteed to converge to the fixed point because it performs stochastic steepest descent. This is essentially the Widrow-Huff algorithm [12,13].

To illustrate Corollary 1, we show the relationships between GRW-I and ML for three different generative models: (i) the $\Delta P$ model, (ii) the noisy-or model, and (iii) the most general form of $P(E|\vec{C})$ for two causes. *It is important to realize that these generative models form a hierarchy and GRW-I algorithms for the later models will also perform ML on the simpler ones.*

**1. The $\Delta P$ model.**

Set $n = 2$, $h_1(\vec{C}) = C_1$ and $h_2(\vec{C}) = C_2$. Then equation (12) reduces to the basic RW algorithm (5) with two weights $V_1, V_2$. By Corollary 1, we see that it performs ML estimation for the $\Delta P$ model (1). This rederives the known relationship between basic RW, ML, and the $\Delta P$ model [4,9].

Observe that Corollary 1 requires that the matrix $\begin{pmatrix} < C_1 >_{P(\vec{C})} & < C_1 C_2 >_{P(\vec{C})} \\ < C_1 C_2 >_{P(\vec{C})} & < C_2 >_{P(\vec{C})} \end{pmatrix}$

be invertible. This is equivalent to the genericity condition $< C_1 C_2 >^2_{P(\vec{C})} \neq < C_1 >_{P(\vec{C})} < C_2 >_{P(\vec{C})}$.

## 2. The Noisy-Or model.

Set $n = 3$ with $h_1(\vec{C}) = C_1, h_2(\vec{C}) = C_2, h_3(\vec{C}) = C_1 C_2$. Then Corollary 1 proves that the following algorithm will converge to estimate $V_1^* = \omega_1, V_2^* = \omega_2$ and $V_3^* = -\omega_1\omega_2$ for the noisy-or model.

$$\begin{aligned}
\Delta V_1 &= C_1(E - C_1 V_1 - C_2 V_2 - C_1 C_2 V_3) = C_1(E - V_1 - C_2 V_2 - C_2 V_3) \\
\Delta V_2 &= C_2(E - C_1 V_1 - C_2 V_2 - C_1 C_2 V_3) = C_2(E - C_1 V_1 - V_2 - C_1 V_3) \\
\Delta V_3 &= C_1 C_2(E - C_1 V_1 - C_2 V_2 - C_1 C_2 V_3) = C_1 C_2(E - V_1 - V_2 - V_3).
\end{aligned} \qquad (13)$$

This algorithm is a minor variant of basic RW. Observe that this has more weights ($n = 3$) than the total number of causes. The first two weights $V_1$ and $V_2$ yield $\omega_1, \omega_2$ while the third weight $V_3$ gives a (redundant) estimate of $\omega_1\omega_2$. The matrix $< h_i(\vec{C})h_j(\vec{C}) >_{P(\vec{C})}$ has determinant $(< C_1 C_2 > - < C_1 >)(< C_1 C_2 > - < C_2 >) < C_1 C_2 >$ and is invertible provided $< C_1 > \neq 0, 1, < C_2 > \neq 0, 1$ and $< C_1 C_2 > \neq < C_1 > < C_2 >$. This rules out the special case in Cheng's experiments [4,5] where $C_1 = 1$ always, see discussion in section (5).

It is known that basic RW is unable to do ML estimation for the noisy-or model *if there are only two weights* [4,5,9,10]. The differences here is that three weights are used.

## 3. The general two-cause model.

Thirdly, we consider the most general model $P(E|\vec{C})$ for two causes. This can be written in the form:

$$P(E = 1|C_1, C_2) = \alpha_1 + \alpha_2 C_1 + \alpha_3 C_2 + \alpha_4 C_1 C_2. \qquad (14)$$

This corresponds to $h_1(\vec{C}) = 1, h_2(\vec{C}) = C_1, h_3(\vec{C}) = C_2, h_4(\vec{C}) = C_1 C_2$. Corollary 1 gives us *the most general algorithm*:

$$\begin{aligned}
\Delta V_1 &= (E - V_1 - C_1 V_2 - C_2 V_3 - C_1 C_2 V_4) = (E - V_1 - C_1 V_2 - C_2 V_3 - C_1 C_2 V_4) \\
\Delta V_2 &= C_1(E - V_1 - C_1 V_2 - C_2 V_3 - C_1 C_2 V_4) = C_1(E - V_1 - V_2 - C_2 V_3 - C_2 V_4) \\
\Delta V_3 &= C_2(E - V_1 - C_1 V_2 - C_2 V_3 - C_1 C_2 V_4) = C_2(E - V_1 - C_1 V_2 - V_3 - C_1 V_4) \\
\Delta V_4 &= C_1 C_2(E - V_1 - C_1 V_2 - C_2 V_3 - C_1 C_2 V_4) = C_1 C_2(E - V_1 - V_2 - V_3 - V_4).
\end{aligned}$$

By Corollary 1, this algorithm will converge to $V_1^* = \alpha_1, V_2^* = \alpha_2, V_3^* = \alpha_3, V_4^* = \alpha_4$, provided the matrix is invertible. The determinant of the matrix $< h_i(\vec{C})h_j(\vec{C}) >_{P(\vec{C})}$ is $< C_1 C_2 > (< C_1 C_2 > - < C_1 >)(< C_1 C_2 > - < C_2 >)(1 - < C_1 > - < C_2 > + < C_1 C_2 >)$. This will be zero for special cases, for example if $C_1 = 1$ always.

*It is important to realize that the most general GRW-I algorithm will converge if $P(E|\vec{C})$ is the $\Delta P$ or the noisy-or model. For $\Delta P$ it will converge to $V_1^* = 0, V_2^* = \omega_1, V_3^* = \omega_2, V_4^* = 0$. For noisy-or, it converges to $V_1^* = 0, V_2^* = \omega_1, V_3^* = \omega_2, V_4^* = -\omega_1\omega_2$.*

The learning system which implements the GRW-I algorithm will not know *a priori* whether the data is generated by $\Delta P$, noisy-or, or the general model for $P(E|C_1, C_2)$. It is therefore better to implement the most general algorithm because this works whatever model generated the data.

Note: other functions $\{h_i(\vec{C})\}$ will lead to different ways to parameterize the probability distribution $P(E|\vec{C})$. They will correspond to different RW algorithms. But their basic properties will be similar to those discussed in this section.

## 4.2 Generalized RW Class II

We can obtain a second class of generalized RW algorithms which perform ML estimation.

**Corollary 2**. *A sufficient condition for RW to have unique fixed point at the ML estimate of the generative model $P(E|\vec{C})$ is that $f_{ij}(\vec{C}) = -g_i(\vec{C})h_j(\vec{C})$, provided the matrix $< h_i(\vec{C})h_j(\vec{C}) >_{P(\vec{C})}$ is invertible.*

Proof. *Direct verification.*

Corollary 2 defines GRW-II to be of form:

$$\Delta V_i = g_i(\vec{C})\{E - \sum_j h_j(\vec{C})V_j\}. \tag{15}$$

We illustrate GRW-II by applying it to the noisy-or model (2). It gives an algorithm very similar to equation (6).

Set $h_1(\vec{C}) = C_1, h_2(\vec{C}) = C_2, h_3(\vec{C}) = C_1C_2$ and $g_1(\vec{C}) = C_1(1 - C_2), g_2(\vec{C}) = C_2(1 - C_1), g_3(\vec{C}) = C_1C_2$.

Corollary 2 yields the update rule:

$$\begin{aligned}
\Delta V_1 &= C_1(1 - C_2)\{E - C_1V_1 - C_2V_2 - C_1C_2V_3\} = C_1(1 - C_2)\{E - V_1\}, \\
\Delta V_2 &= C_2(1 - C_1)\{E - C_1V_1 - C_2V_2 - C_1C_2V_3\} = C_2(1 - C_1)\{E - V_2\}, \\
\Delta V_3 &= C_1C_2\{E - C_1V_1 - C_2V_2 - C_1C_2V_3\} = C_1C_2\{E - V_1 - V_2 - V_3\}.
\end{aligned} \tag{16}$$

The matrix $< h_i(\vec{C})h_j(\vec{C}) >_{P(\vec{C})}$ has determinant $< C_1C_2 > (< C_1 > - < C_1C_2 >)(< C_2 > - < C_1C_2 >)$ and so is invertible for generic $P(\vec{C})$. The algorithm will converge to weights $V_1^* = \omega_1, V_2^* = \omega_2, V_3^* = -\omega_1\omega_2$. If we change the model to $\Delta P$, then we get convergence to $V_1^* = \omega_1, V_2^* = \omega_2, V_3^* = 0$.

Observe that the equations (16) are largely decoupled. In particular, the updates for $V_1$ and $V_2$ do not depend on the third weight $V_3$. It is possible to remove the update equation for $V_3$ by setting $g_3(\vec{C}) = 0$. The remaining update equations for $V_1 \& V_2$ will converge to $\omega_1, \omega_2$ for both the noisy-or and the $\Delta P$ model.

These reduced update equations are identical to those given by equation (6) which were proven to converge to $\omega_1, \omega_2$ [10]. We note that the matrix $< h_i(\vec{C})h_j(\vec{C}) >_{P(\vec{C})}$ now has a zero eigenvalue (because $g_3(\vec{C}) = 0$) but this does not matter because it corresponds to the third weight $V_3$. The matrix remains invertible if we restrict it to $i, j = 1, 2$.

A limitation of GRW-II algorithm of equation (16) is that it only updates the weights if only one cause is active. So it would fail to explain effects such as blocking where both causes are on for part of the stimuli (Dayan personal communication).

## 5  Non-generic, coordinate transformations, and non-linear RW

Our results have assumed genericity constraints on the distribution $P(\vec{C})$ of causes. They usually correspond to cases where one cause is always present. We now briefly discuss what happens when these constraints are violated. For simplicity, we concentrate on an important special case.

Cheng's PC theory [4,5] uses the noisy-or model for generating the data but cause $C_1$ is a background cause which is on all the time (i.e. $C_1 = 1$ always). This implies that

$< C_2 >=< C_1 C_2 >$ and so we cannot apply RW algorithms (13), the most general algorithm, or (16) because the matrix determinant will be zero in all three cases. Since $C_1 = 1$ we can drop it as a variable and re-express the noisy-or model as:

$$P(E = 1|\vec{C}) = \omega_1 + \omega_2(1 - \omega_1)C_2. \tag{17}$$

Theorem 1 shows that we can define generalized RW algorithms to find ML estimates of $\omega_1$ and $\omega_2(1 - \omega_1)$ (assuming $\omega_1 \neq 1$). But, conversely, it is impossible to estimate $\omega_2$ directly by any linear generalized RW.

The problem is simply a matter of different coordinate systems. RW estimates the parameters of the generative model in a different coordinate system than the one used to specify the model. There is a non-linear transformation between the coordinates systems relating $\{\omega_1, \omega_2\}$ to $\{\omega_1, \omega_2(1 - \omega_1)\}$. So RW can estimate the ML parameters provided we allow for an additional non-linear transformation. From this perspective, the inability to RW to perfrom ML estimation for Cheng's model is merely an artifact. If we reparameterize the generative model to be $P(E = 1|\vec{C}) = \omega_1 + \hat{\omega}_2 C_2$, where $\hat{\omega}_2 = \omega_2(1 - \omega_1)$, then we can design an RW to estimate $\{\omega_1, \hat{\omega}_2\}$.

The non-linear transformation breaks down if $\omega_1 = 1$. In this case, the generative model $P(E|\vec{C})$ becomes independent of $\omega_2$ and so it is impossible to estimate it.

But suppose we want to really estimate $\omega_1$ and $\omega_2$ directly (for Cheng's model, the value of $\omega_2$ is the causal power and hence is a meaningful quantity [4,5]). To do this we first define a linear RW to estimate $\omega_1$ and $\hat{\omega}_2 = \omega_2(1 - \omega_1)$. The equations are:

$$V_1^{t+1} = V_1^t + \gamma_1 \Delta V_1^t, \quad V_2^{t+1} = V_2^t + \gamma_2 \Delta V_2^t. \tag{18}$$

with $< V_1 > \mapsto \omega_1$ and $< V_2 > \mapsto \omega_2$ for large $t$. The fluctuations (variances) are scaled by the parameters $\gamma_1, \gamma_2$ and hence can be made arbitrarily small, see [10].

To estimate $\omega_2$, we replace the variable $V_2$ by a new variable $V_3 = V_2/(1 - V_1)$ which is updated by a nonlinear equation ($V_1$ is updated as before):

$$V_3^{t+1} = V_3^t + \frac{V_3^t}{1 - V_1^t}\delta V_1^t + \frac{\Delta V_2^t}{1 - V_1^t}, \tag{19}$$

where we use $V_3 = V_2/(1 - V_1)$ to re-express $\Delta V_1$ and $\Delta V_2$ in terms of functions of $V_1$ and $V_3$. Provided the fluctuations are small, by controlling the size of the $\gamma$'s, we can ensure that $V_3$ converges arbitrarily close to $\hat{\omega}_2/(1 - \omega_1) = \omega_2$.

## 6 Conclusion

This paper shows that we can obtain linear generalizations of the Rescorla-Wagner algorithm which can learn the parameters of generative models by Maximum Likelihood. For one class of RW generalizations we have only shown that the fixed points are unique and correspond to ML estimates of the parameters of the generative models. But Theorems 3,4 & 5 of Yuille (2004) can be applied to determine convergence conditions. Convergence rates can be determined by these Theorems provided that the data is generated as i.i.d. samples from the generative model. These theorems can also be used to obtain convergence results for piecewise i.i.d. samples as occurs in forward and backward blocking experiments.

These generalizations of Rescorla-Wagner require augmenting the number of weight variables. This was already proposed, on experimental grounds, so that new weights get created if causes occur in conjunction, [2]. Note that this happens naturally in the algorithms presented (13, the most general algorithm,16) – weights remain at zero until we get an event

$C_1 C_2 = 1$. It is straightforward to extend the analysis to models with conjunctions of many causes. We conjecture that these generalizations converge to good approaximation to ML estimates if we truncate the conjunction of causes at a fixed order.

Finally, many of our results have involved a genericity assumption on the distribution of causes $P(\vec{C})$. We have argued that when these assumptions are violated, for example in Cheng's experiments, then generalized RW still performs ML estimation, but with a non-linear transform. Alternatively we have shown how to define a nonlinear RW that estimates the parameters directly.

## Acknowledgement

I acknowledge helpful conversations with Peter Dayan, Rich Shiffrin, and Josh Tennenbaum. I thank Aaron Courville for describing augmented Rescorla-Wagner. I thank the W.M. Keck Foundation for support and NSF grant 0413214.

## References

[1]. R.A. Rescorla and A.R. Wagner. "A Theory of Pavlovian Conditioning". In A.H. Black andW.F. Prokasy, eds. **Classical Conditioning II: Current Research and Theory.** New York. Appleton-Century-Crofts, pp 64-99. 1972.

[2] R.A. Rescorla. Journal of Comparative and Physiological Psychology. 79, 307. 1972.

[3]. B. A. Spellman. "Conditioning Causality". In D.R. Shanks, K.J. Holyoak, and D.L. Medin, (eds). **Causal Learning: The Psychology of Learning and Motivation, Vol. 34**. San Diego, California. Academic Press. pp 167-206. 1996.

[4]. P. Cheng. "From Covariance to Causation: A Causal Power Theory". *Psychological Review*, **104**, pp 367-405. 1997.

[5]. M. Buehner and P. Cheng. "Causal Induction: The power PC theory versus the Rescorla-Wagner theory". In *Proceedings of the 19th Annual Conference of the Cognitive Science Society"*. 1997.

[6]. J.B. Tenenbaum and T.L. Griffiths. "Structure Learning in Human Causal Induction". Advances in Neural Information Processing Systems 12. MIT Press. 2001.

[7]. D. Danks, T.L. Griffiths, J.B. Tenenbaum. "Dynamical Causal Learning". *Advances in Neural Information Processing Systems 14*. 2003.

[8] A.C. Courville, N.D. Dew, and D.S. Touretsky. "Similarity and discrimination in classical conditioning". NIPS. 2004.

[9]. D. Danks. "Equilibria of the Rescorla-Wagner Model". *Journal of Mathematical Psychology*. Vol. 47, pp 109-121. 2003.

[10] A.L. Yuille. "The Rescorla-Wagner algorithm and Maximum Likelihood estimation of causal parameters". NIPS. 2004.

[11]. P. Dayan and S. Kakade. "Explaining away in weight space". In *Advances in Neural Information Processing Systems 13*. 2001.

[12] B. Widrow and M.E. Hoff. "Adapting Switching Circuits". *1960 IRE WESCON Conv. Record.*, Part 4, pp 96-104. 1960.

[13] A.G. Barto and R.S. Sutton. "Time-derivative Models of Pavlovian Conditioning". In *Learning and Computational Neuroscience: Foundations of Adaptive Networks*. M. Gabriel and J. Moore (eds). pp 497-537. MIT Press. Cambridge, MA. 1990.
